# Agnostic PAC-Learning of Functions on Analog Neural Nets

## (Extended Abstract)

**Wolfgang Maass**

Institute for Theoretical Computer Science
Technische Universitaet Graz
Klosterwiesgasse 32/2
A-8010 Graz, Austria
e-mail: maass@igi.tu-graz.ac.at

**Abstract:**

There exist a number of negative results ([J], [BR], [KV]) about learning on neural nets in Valiant's model [V] for probably approximately correct learning ("PAC-learning"). These negative results are based on an asymptotic analysis where one lets the number of nodes in the neural net go to infinity. Hence this analysis is less adequate for the investigation of learning on a small *fixed* neural net with relatively few *analog* inputs (e.g. the principal components of some sensory data). The latter type of learning problem gives rise to a different kind of asymptotic question: Can the true error of the neural net be brought arbitrarily close to that of a neural net with "optimal" weights through sufficiently long training? In this paper we employ some new arguments in order to give a positive answer to this question in Haussler's rather realistic refinement of Valiant's model for PAC-learning ([H], [KSS]). In this more realistic model no a-priori assumptions are required about the "learning target", noise is permitted in the training data, and the inputs and outputs are not restricted to boolean values. As a special case our result implies one of the first positive results about learning on multi-layer neural nets in Valiant's original PAC-learning model. At the end of this paper we will describe an efficient *parallel* implementation of this new learning algorithm.

We consider multi-layer high order feedforward neural nets $\mathcal{N}$ with arbitrary piecewise polynomial activation functions. Each node $g$ of fan-in $m > 0$ in $\mathcal{N}$ is called a *computation node*. It is labelled by some polynomial $Q^g(y_1, \ldots, y_m)$ and some piecewise polynomial activation function $\gamma^g : \mathbf{R} \to \mathbf{R}$. We assume that $\gamma^g$ consists of finitely many polynomial pieces and that its definition involves only rational parameters. The computation node $g$ computes the function $\langle y_1, \ldots, y_m \rangle \mapsto \gamma^g(Q^g(y_1, \ldots, y_m))$ from $\mathbf{R}^m$ into $\mathbf{R}$. The nodes of fan-in 0 in $\mathcal{N}$ ("input nodes") are labelled by variables $x_1, \ldots, x_k$. The nodes $g$ of fan-out 0 in $\mathcal{N}$ ("output nodes") are labelled by $1, \ldots, l$. We assume that the range $B$ of their activation functions $\gamma^g$ is bounded. Any parameters that occur in the definitions of the $\gamma^g$ are referred to as *architectural parameters* of $\mathcal{N}$.

The coefficients of all the polynomials $Q^g$ are called the *programmable parameters* (or *weights*) of $\mathcal{N}$. Let $w$ be the number of programmable parameters of $\mathcal{N}$. For any assignment $\underline{\alpha} \in \mathbf{R}^w$ to the programmable parameters of $\mathcal{N}$ the network computes a function from $\mathbf{R}^k$ into $\mathbf{R}^l$ which we will denote by $\mathcal{N}^{\underline{\alpha}}$.

We write $\mathbf{Q}_n$ for the set of rational numbers that can be written as quotients of integers with bit-length $\leq n$. For $\underline{z} = \langle z_1, \ldots, z_l \rangle \in \mathbf{R}^l$ we write $||\underline{z}||_1$ for $\sum_{j=1}^{l} |z_j|$.

Let $F : \mathbf{R}^k \to \mathbf{R}^l$ be some arbitrary function, which we will view as a "prediction rule". For any given instance $\langle \underline{x}, \underline{y} \rangle \in \mathbf{R}^k \times \mathbf{R}^l$ we measure the *error* of $F$ by $||F(\underline{x}) - \underline{y}||_1$. For any distribution $A$ over some subset of $\mathbf{R}^k \times \mathbf{R}^l$ we measure the *true error of $F$ with regard to $A$* by $E_{\langle \underline{x}, \underline{y} \rangle \in A}[||F(\underline{x}) - \underline{y}||_1]$, i.e. the expected value of the error of $F$ with respect to distribution $A$.

**Theorem 1:** *Let $\mathcal{N}$ be some arbitrary high order feedforward neural net with piecewise polynomial activation functions. Let $w$ be the number of programmable parameters of $\mathcal{N}$ (we assume that $w = O(1)$). Then one can construct from $\mathcal{N}$ some first order feedforward neural net $\tilde{\mathcal{N}}$ with piecewise linear activation functions and the quadratic activation function $\gamma(x) = x^2$, which has the following property:*
*There exists a polynomial $m(\frac{1}{\varepsilon}, \frac{1}{\delta})$ and a learning algorithm LEARN such that for any given $\varepsilon, \delta, \in (0, 1)$ and $s, n \in N$ and any distribution $A$ over $Q_n^k \times (Q_n \cap B)^l$ the following holds:*
*For any sample $\zeta = (\langle \underline{x}_i, \underline{y}_i \rangle)_{i=1,\ldots,m}$ of $m \geq m(\frac{1}{\varepsilon}, \frac{1}{\delta})$ points that are independently drawn according to $A$ the algorithm LEARN computes in polynomially in $m, s, n$ computation steps an assignment $\tilde{\underline{\alpha}}$ of rational numbers to the programmable parameters of $\tilde{\mathcal{N}}$ such that with probability $\geq 1 - \delta$:*

$$E_{\langle \underline{x}, \underline{y} \rangle \in A}[||\tilde{\mathcal{N}}^{\tilde{\underline{\alpha}}}(\underline{x}) - \underline{y}||_1] \leq \varepsilon + \inf_{\underline{\alpha} \in Q_s^w} E_{\langle \underline{x}, \underline{y} \rangle \in A}[||\mathcal{N}^{\underline{\alpha}}(\underline{x}) - \underline{y}||_1],$$

*or in other words:*
*The true error of $\tilde{\mathcal{N}}^{\tilde{\underline{\alpha}}}$ with regard to $A$ is within $\varepsilon$ of the least possible true error that can be achieved by any $\mathcal{N}^{\underline{\alpha}}$ with $\underline{\alpha} \in Q_s^w$.*

**Remarks**
   a) One can easily see (see [M 93b] for details) that Theorem 1 provides a positive learning result in Haussler's extension of Valiant's model for PAC-learning ([H], [KSS]). The "touchstone class" (see [KSS]) is defined as the

class of function $f : \mathbf{R}^k \to \mathbf{R}^l$ that are computable on $\mathcal{N}$ with programmable parameters from $\mathbf{Q}$.

This fact is of some general interest, since so far only very few positive results are known for *any* learning problem in this rather realistic (but quite demanding) learning model.

b) Consider the special case where the distribution $A$ over $\mathbf{Q}_n^k \times (\mathbf{Q}_n \cap B)^l$ is of the form

$$A_{D,\underline{\alpha}_T}(\underline{x}, \underline{y}) = \begin{cases} D(\underline{x}) & , \quad \text{if } \underline{y} = \mathcal{N}^{\underline{\alpha}_T}(\underline{x}) \\ 0 & , \quad \text{otherwise} \end{cases}$$

for some *arbitrary* distribution $D$ over the domain $\mathbf{Q}_n^k$ and some *arbitrary* $\underline{\alpha}_T \in \mathbf{Q}_s^w$. Then the term

$$\inf_{\underline{\alpha} \in \mathbf{Q}_s^w} E_{(\underline{x},\underline{y}) \in A}[||\mathcal{N}^{\underline{\alpha}}(\underline{x}) - \underline{y}||_1]$$

is equal to 0. Hence the preceding theorem states that with learning algorithm LEARN the "learning network" $\tilde{\mathcal{N}}$ can "learn" with arbitrarily small true error any target function $\mathcal{N}^{\underline{\alpha}_T}$ that is computable on $\mathcal{N}$ with rational "weights" $\underline{\alpha}_T$. Thus by choosing $\mathcal{N}$ sufficiently large, one can guarantee that the associated "learning network" $\tilde{\mathcal{N}}$ can learn any target-function that might arise in the context of a specific learning problem.

In addition the theorem also applies to the more realistic situation where the learner receives examples $\langle \underline{x}, \underline{y} \rangle$ of the form $\langle \underline{x}, \mathcal{N}^{\underline{\alpha}_T}(\underline{x}) + \text{noise} \rangle$, or even if there exists no "target function" $\mathcal{N}^{\underline{\alpha}_T}$ that would "explain" the actual distribution $A$ of examples $\langle \underline{x}, \underline{y} \rangle$ ("agnostic learning").

The **proof** of Theorem 1 is mathematically quite involved, and we can give here only an outline. It consists of three steps:

(1) Construction of the auxiliary neural net $\tilde{\mathcal{N}}$.

(2) Reducing the optimization of weights in $\tilde{\mathcal{N}}$ for a given distribution $A$ to a *finite* nonlinear optimization problem.

(3) Reducing the resulting finite nonlinear optimization problem to a family of finite *linear* optimization problems.

**Details to step (1):** If the activation functions $\gamma^g$ in $\mathcal{N}$ are piecewise linear and all computation nodes in $\mathcal{N}$ have fan-out $\leq 1$ (this occurs for example if $\mathcal{N}$ has just one hidden layer and only one output) then one can set $\tilde{\mathcal{N}} := \mathcal{N}$. If the $\gamma^g$ are piecewise linear but not all computation nodes in $\mathcal{N}$ have fan-out $\leq 1$ one defines $\tilde{\mathcal{N}}$ as the tree of the same depth as $\mathcal{N}$, where subcircuits of computation nodes with fan-out $m > 1$ are duplicated $m$ times. The activation functions remain unchanged in this case.

If the activation functions $\gamma^g$ are piecewise polynomial but not piecewise linear, one has to apply a rather complex construction which is described in detail in the Journal version of [M 93a]. In any case $\tilde{\mathcal{N}}$ has the property that all functions that

are computable on $\mathcal{N}$ can also be computed on $\tilde{\mathcal{N}}$, the depth of $\tilde{\mathcal{N}}$ is bounded by a constant, and the size of $\tilde{\mathcal{N}}$ is bounded by a polynomial in the size of $\mathcal{N}$ (provided that the depth and order of $\mathcal{N}$, as well as the number and degrees of the polynomial pieces of the $\gamma^g$ are bounded by a constant).

**Details to step (2):** Since the VC-dimension of a neural net is only defined for neural nets with *boolean* output, one has to consider here instead the *pseudo-dimension* of the function class $\mathcal{F}$ that is defined by $\tilde{\mathcal{N}}$.

**Definition:** *(see Haussler [H]).*
*Let $X$ be some arbitrary domain, and let $\mathcal{F}$ be an arbitrary class of functions from $X$ into $\mathbf{R}$. Then the pseudo-dimension of $\mathcal{F}$ is defined by*

$$\dim_P(\mathcal{F}) := \max\{|S| : S \subseteq X \text{ and } \exists h : S \to \mathbf{R} \text{ such that}$$
$$\forall b \in \{0,1\}^S \, \exists f \in \mathcal{F} \, \forall x \in S \, (f(x) \geq h(x) \Leftrightarrow b(x) = 1)\}.$$

Note that in the special case where $\mathcal{F}$ is a concept class (i.e. all $f \in \mathcal{F}$ are $0-1$ valued) the pseudo-dimension $\dim_P(\mathcal{F})$ coincides with the VC-dimension of $\mathcal{F}$. The pseudo-dimension of the function class associated with network architectures $\tilde{\mathcal{N}}$ with piecewise polynomial activation functions can be bounded with the help of Milnor's Theorem [Mi] in the same way as the VC-dimension for the case of boolean network output (see [GJ]):

**Theorem 2:** *Consider arbitrary network architectures $\tilde{\mathcal{N}}$ of order $v$ with $k$ input nodes, $l$ output nodes, and $w$ programmable parameters. Assume that each gate in $\tilde{\mathcal{N}}$ employs as activation function some piecewise polynomial (or piecewise rational) function of degree $\leq d$ with at most $q$ pieces. For some arbitrary $p \in \{1,2,\ldots\}$ we define $\mathcal{F} := \{ f : \mathbf{R}^{k+l} \to \mathbf{R} : \exists \underline{\alpha} \in \mathbf{R}^w \, \forall \underline{x} \in \mathbf{R}^k \, \forall \underline{y} \in \mathbf{R}^l (f(\underline{x},\underline{y}) = ||\tilde{\mathcal{N}}^{\underline{\alpha}}(\underline{x}) - \underline{y}||_p)\}$. Then one has $\dim_P(\mathcal{F}) = O(w^2 \log q)$ if $v,d,l = O(1)$.* ∎

With the help of the pseudo-dimension one can carry out the desired reduction of the optimization of weights in $\tilde{\mathcal{N}}$ (with regard to an arbitrary given distribution $A$ of examples $\langle \underline{x}, \underline{y} \rangle$) to a *finite* optimization problem. Fix some interval $[b_1,b_2] \subseteq \mathbf{R}$ such that $B \subseteq [b_1,b_2], b_1 < b_2$, and such that the ranges of the activation functions of the output gates of $\tilde{\mathcal{N}}$ are contained in $[b_1,b_2]$. We define $b := l \cdot (b_2 - b_1)$, and $\mathcal{F} := \{f : \mathbf{R}^k \times [b_1,b_2]^l \to [0,b] : \exists \underline{\alpha} \in \mathbf{R}^w \, \forall \underline{x} \in \mathbf{R}^k \, \forall \underline{y} \in [b_1,b_2]^l \, (f(\underline{x},\underline{y}) = ||\tilde{\mathcal{N}}^{\underline{\alpha}}(\underline{x}) - \underline{y}||_1)\}$. Assume now that parameters $\varepsilon, \delta \in (0,1)$ with $\varepsilon \leq b$ and $s,n \in \mathbf{N}$ have been fixed. For convenience we assume that $s$ is sufficiently large so that all architectural parameters in $\mathcal{N}$ are from $\mathbf{Q}_s$ (we assume that all architectural parameters in $\mathcal{N}$ are rational). We define

$$m\left(\frac{1}{\varepsilon}, \frac{1}{\delta}\right) := \frac{257 \cdot b^2}{\varepsilon^2}\left(2 \cdot \dim_P(\mathcal{F}) \cdot ln\frac{33eb}{\varepsilon} + ln\frac{8}{\delta}\right).$$

By Corollary 2 of Theorem 7 in Haussler [H] one has for $m \geq m(\frac{1}{\varepsilon}, \frac{1}{\delta})$, $K := \frac{\sqrt{257}}{8} \in (2,3)$, and any distribution $A$ over $\mathbf{Q}_n^k \times (\mathbf{Q}_n \cap [b_1,b_2])^l$

$$(1) \quad Pr_{\zeta \in A^m}[\{\exists f \in \mathcal{F} : |(\frac{1}{m}\sum_{(\underline{x},\underline{y}) \in \zeta} f(\underline{x},\underline{y})) - E_{(\underline{x},\underline{y}) \in A}[f(\underline{x},\underline{y})]| > \frac{\varepsilon}{K}\}] \leq \delta,$$

where $E_{(\underline{x},\underline{y})\in A}[f(\underline{x},\underline{y})]$ is the expectation of $f(\underline{x},\underline{y})$ with regard to distribution $A$.

We design an algorithm LEARN that computes for any $m \in \mathbf{N}$, any sample

$$\zeta = (\langle \underline{x}_i, \underline{y}_i \rangle)_{i \in \{1,\dots,m\}} \in (\mathbf{Q}_n^k \times (\mathbf{Q}_n \cap [b_1, b_2])^l)^m,$$

and any given $s \in \mathbf{N}$ in polynomially in $m, s, n$ computation steps an assignment $\tilde{\underline{\alpha}}$ of rational numbers to the parameters in $\tilde{\mathcal{N}}$ such that the function $\tilde{h}$ that is computed by $\tilde{\mathcal{N}}^{\tilde{\underline{\alpha}}}$ satisfies

$$(2) \quad \frac{1}{m}\sum_{i=1}^{m} ||\tilde{h}(\underline{x}_i) - \underline{y}_i||_1 \leq (1 - \frac{2}{K})\varepsilon + \inf_{\underline{\alpha} \in \mathbf{Q}_s^w} \frac{1}{m}\sum_{i=1}^{m} ||\mathcal{N}^{\underline{\alpha}}(\underline{x}_i) - \underline{y}_i||_1.$$

This suffices for the proof of Theorem 1, since (1) and (2) together imply that, for any distribution $A$ over $\mathbf{Q}_n^k \times (\mathbf{Q}_n \cap [b_1, b_2])^l$ and any $m \geq m(\frac{1}{\varepsilon}, \frac{1}{\delta})$, with probability $\geq 1 - \delta$ (with respect to the random drawing of $\zeta \in A^m$) the algorithm LEARN outputs for inputs $\zeta$ and $s$ an assignment $\tilde{\underline{\alpha}}$ of rational numbers to the parameters in $\tilde{\mathcal{N}}$ such that

$$E_{(\underline{x},\underline{y})\in A}[||\tilde{\mathcal{N}}^{\tilde{\underline{\alpha}}}(\underline{x}) - \underline{y}||_1] \leq \varepsilon + \inf_{\underline{\alpha} \in \mathbf{Q}_s^w} E_{(\underline{x},\underline{y})\in A}[||\mathcal{N}^{\underline{\alpha}}(\underline{x}) - \underline{y}||_1].$$

**Details to step (3):** The computation of weights $\tilde{\underline{\alpha}}$ that satisfy (2) is nontrivial, since this amounts to solving a *nonlinear* optimization problem. This holds even if each activation function in $\tilde{\mathcal{N}}$ is piecewise *linear*, because weights from successive layers are multiplied with each other.

We employ a method from [M 93a] that allows us to replace the nonlinear conditions on the programmable parameters $\underline{\alpha}$ of $\tilde{\mathcal{N}}$ by linear conditions for a transformed set $\underline{c}, \underline{\beta}$ of parameters. We simulate $\tilde{\mathcal{N}}^{\underline{\alpha}}$ by another network architecture $\hat{\mathcal{N}}[\underline{c}]^{\underline{\beta}}$ (which one may view as a "normal form" for $\tilde{\mathcal{N}}^{\underline{\alpha}}$) that uses the same graph $\langle V, E \rangle$ as $\tilde{\mathcal{N}}$, but different activation functions and different values $\underline{\beta}$ for its programmable parameters. The activation functions of $\hat{\mathcal{N}}[\underline{c}]$ depend on $|V|$ new architectural parameters $\underline{c} \in \mathbf{R}^{|V|}$, which we call *scaling parameters* in the following. Whereas the architectural parameters of a network architecture are usually kept fixed, we will be forced to change the scaling parameters of $\hat{\mathcal{N}}$ along with its programmable parameters $\underline{\beta}$. Although this new network architecture has the *disadvantage* that it requires $|V|$ additional parameters $\underline{c}$, it has the *advantage* that we can choose in $\hat{\mathcal{N}}[\underline{c}]$ all weights on edges *between* computation nodes to be from $\{-1, 0, 1\}$. Hence we can treat them as constants with at most 3 possible values in the system of inequalities that describes computations of $\hat{\mathcal{N}}[\underline{c}]$. Thereby we can achieve that all variables that appear in the inqualities that describe computations of $\hat{\mathcal{N}}[\underline{c}]$ for fixed network inputs (the variables for weights of gates on level 1, the variables for the biases of gates on all levels, *and the new variables for the scaling parameters $\underline{c}$*) appear only *linearly* in those inqualities.

We briefly indicate the construction of $\hat{\mathcal{N}}$ in the case where each activation function $\gamma$ in $\tilde{\mathcal{N}}$ is piecewise linear. For any $c > 0$ we consider the associated piecewise linear activation function $\gamma^c$ with

$$\forall x \in \mathbf{R}(\gamma^c(c \cdot x) = c \cdot \gamma(x)).$$

Assume that $\underline{\alpha}$ is some arbitrary given assignment to the programmable parameters in $\tilde{\mathcal{N}}$. We transform $\tilde{\mathcal{N}}^{\underline{\alpha}}$ through a recursive process into a "normal form" $\hat{\mathcal{N}}[\underline{c}]^{\underline{\beta}}$ in which all weights on edges between computation nodes are from $\{-1, 0, 1\}$, such that $\forall \underline{x} \in \mathbf{R}^k \left( \tilde{\mathcal{N}}^{\underline{\alpha}}(\underline{x}) = \hat{\mathcal{N}}[\underline{c}]^{\underline{\beta}}(\underline{x}) \right)$.

Assume that an output gate $g_{out}$ of $\tilde{\mathcal{N}}^{\underline{\alpha}}$ receives as input $\sum_{i=1}^{q} \alpha_i y_i + \alpha_0$, where $\alpha_1, \ldots, \alpha_q, \alpha_0$ are the weights and the bias of $g_{out}$ (under the assignment $\underline{\alpha}$) and $y_1, \ldots, y_q$ are the (real valued) outputs of the immediate predecessors $g_1, \ldots, g_q$ of $g$. For each $i \in \{1, \ldots, q\}$ with $\alpha_i \neq 0$ such that $g_i$ is not an input node we replace the activation function $\gamma_i$ of $g_i$ by $\gamma_i^{|\alpha_i|}$, and we multiply the weights and the bias of gate $g_i$ with $|\alpha_i|$. Finally we replace the weight $\alpha_i$ of gate $g_{out}$ by $\mathrm{sgn}(\alpha_i)$, where $\mathrm{sgn}(\alpha_i) := 1$ if $\alpha_i > 0$ and $\mathrm{sgn}(\alpha_i) := -1$ if $\alpha_i < 0$. This operation has the effect that the multiplication with $|\alpha_i|$ is carried out *before* the gate $g_i$ (rather than after $g_i$, as done in $\tilde{\mathcal{N}}^{\underline{\alpha}}$), but that the considered output gate $g_{out}$ still receives the same input as before. If $\alpha_i = 0$ we want to "freeze" that weight at 0. This can be done by deleting $g_i$ and all gates below $g_i$ from $\hat{\mathcal{N}}$.

The analogous operations are recursively carried out for the predecessors $g_i$ of $g_{out}$ (note however that the weights of $g_i$ are no longer the original ones from $\tilde{\mathcal{N}}^{\underline{\alpha}}$, since they have been changed in the preceding step). We exploit here the assumption that each gate in $\tilde{\mathcal{N}}$ has fan-out $\leq 1$.

Let $\underline{\beta}$ consist of the new weights on edges adjacent to input nodes and of the resulting biases of all gates in $\hat{\mathcal{N}}$. Let $\underline{c}$ consist of the resulting scaling parameters at the gates of $\hat{\mathcal{N}}$. Then we have $\forall \underline{x} \in \mathbf{R}^k \left( \tilde{\mathcal{N}}^{\underline{\alpha}}(\underline{x}) = \hat{\mathcal{N}}[\underline{c}]^{\underline{\beta}}(\underline{x}) \right)$. Furthermore $c > 0$ for all scaling parameters $c$ in $\underline{c}$.

At the end of this proof we will also need the fact that the previously described parameter transformation can be inverted, i.e. one can compute from $\underline{c}, \underline{\beta}$ an equivalent weight assignment $\underline{\alpha}$ for $\tilde{\mathcal{N}}$ (with the *original* activation functions $\gamma$).

We now describe how the algorithm LEARN computes for any given sample $\zeta = (\langle \underline{x_i}, \underline{y_i} \rangle)_{i=1,\ldots,m} \in (\mathbf{Q}_n^k \times (\mathbf{Q}_n \cap [b_1, b_2])^l)^m$ and any given $s \in \mathbf{N}$ with the help of linear programming a new assignment $\tilde{\underline{c}}, \tilde{\underline{\beta}}$ to the parameters in $\hat{\mathcal{N}}$ such that the function $\tilde{h}$ that is computed by $\hat{\mathcal{N}}[\tilde{\underline{c}}]^{\tilde{\underline{\beta}}}$ satisfies (2). For that purpose we describe the computations of $\hat{\mathcal{N}}$ for the *fixed* inputs $\underline{x_i}$ from the sample $\zeta = (\langle \underline{x_i}, \underline{y_i} \rangle)_{i=1,\ldots,m}$ by polynomially in $m$ many systems $L_1, \ldots, L_{p(m)}$ that each consist of $O(m)$ linear inequalities with the transformed parameters $\underline{c}, \underline{\beta}$ as variables. Each system $L_j$ reflects one possibility for employing specific linear pieces of the activation functions in $\hat{\mathcal{N}}$ for specific network inputs $\underline{x_1}, \ldots, \underline{x_m}$, and for employing different combinations of weights from $\{-1, 0, 1\}$ for edges between computation nodes.

One can show that it suffices to consider only polynomially in $m$ many systems of inequalities $L_j$ by exploiting that all inequalities are linear, and that the input space for $\hat{\mathcal{N}}$ has bounded dimension $k$.

We now expand each of the systems $L_j$ (which has only $O(1)$ variables) into a linear programming problem $LP_j$ with $O(m)$ variables. We add to $L_j$ for each of the $l$ output nodes $\nu$ of $\hat{\mathcal{N}}$ $2m$ new variables $u_i^\nu, v_i^\nu$ for $i = 1, \ldots, m$, and the $4m$ inequalities

$$t_j^\nu(\underline{x_i}) \leq (\underline{y_i})_\nu + u_i^\nu - v_i^\nu, \quad t_j^\nu(\underline{x_i}) \geq (\underline{y_i})_\nu + u_i^\nu - v_i^\nu, \quad u_i^\nu \geq 0, \quad v_i^\nu \geq 0,$$

where $(\langle \underline{x_i}, \underline{y_i} \rangle)_{i=1,\ldots,m}$ is the fixed sample $\zeta$ and $(\underline{y_i})_\nu$ is that coordinate of $\underline{y_i}$ which corresponds to the output node $\nu$ of $\hat{\mathcal{N}}$. In these inequalities the symbol $t_j^\nu(\underline{x_i})$ denotes the term (which is by construction linear in the variables $\underline{c}, \underline{\beta}$) that represents the output of gate $\nu$ for network input $\underline{x_i}$ in this system $L_j$. One should note that these terms $t_j^\nu(\underline{x_i})$ will in general be different for different $j$, since different linear pieces of the activation functions at preceding gates may be used in the computation of $\hat{\mathcal{N}}$ for the same network input $\underline{x_i}$. We expand the system $L_j$ of linear inequalities to a linear programming problem $\overline{LP}_j$ in canonical form by adding the optimization requirement

$$\text{minimize} \qquad \sum_{i=1}^m \sum_{\nu \text{ output node}} (u_i^\nu + v_i^\nu).$$

The algorithm LEARN employs an efficient algorithm for linear programming (e.g. the ellipsoid algorithm, see [PS]) in order to compute in altogether polynomially in $m, s$ and $n$ many steps an optimal solution for each of the linear programming problems $LP_1, \ldots, LP_{p(m)}$. We write $h_j$ for the function from $\mathbf{R}^k$ into $\mathbf{R}^l$ that is computed by $\hat{\mathcal{N}}[\underline{c}]^{\underline{\beta}}$ for the optimal solution $\underline{c}, \underline{\beta}$ of $LP_j$. The algorithm LEARN computes $\frac{1}{m} \sum_{i=1}^m \|h_j(\underline{x_i}) - \underline{y_i}\|_1$ for $j = 1, \ldots, p(m)$. Let $\tilde{j}$ be that index for which this expression has a minimal value. Let $\underline{\tilde{c}}, \underline{\tilde{\beta}}$ be the associated optimal solution of $LP_{\tilde{j}}$ (i.e. $\hat{\mathcal{N}}[\underline{\tilde{c}}]^{\underline{\tilde{\beta}}}$ computes $h_{\tilde{j}}$). LEARN employs the previously mentioned backwards transformation from $\underline{\tilde{c}}, \underline{\tilde{\beta}}$ into values $\underline{\tilde{\alpha}}$ for the programmable parameters of $\tilde{\mathcal{N}}$ such that $\forall \underline{x} \in \mathbf{R}^k (\tilde{\mathcal{N}}^{\underline{\tilde{\alpha}}}(\underline{x}) = \hat{\mathcal{N}}[\underline{\tilde{c}}]^{\underline{\tilde{\beta}}}(\underline{x}))$. These values $\underline{\tilde{\alpha}}$ are given as output of the algorithm LEARN.

We refer to [M 93b] for the verification that this weight assignment $\underline{\tilde{\alpha}}$ has the property that is claimed in Theorem 1. We also refer to [M 93b] for the proof in the more general case where the activation functions of $\mathcal{N}$ are piecewise *polynomial*. ∎

**Remark:** The algorithm LEARN can be speeded up substantially on a parallel machine. Furthermore if the individual processors of the parallel machine are allowed to use random bits, hardly any global control is required for this parallel computation. We use polynomially in $m$ many processors. Each processor picks at random one of the systems $L_j$ of linear inequalities and solves the corresponding linear programming problem $LP_j$. Then the parallel machine compares in a "competitive phase" the costs $\sum_{i=1}^m \|h_j(\underline{x_i}) - \underline{y_i}\|_1$ of the solutions $h_j$ that have been computed by the individual processors. It outputs the weights $\underline{\tilde{\alpha}}$ for $\tilde{\mathcal{N}}$ that correspond to the

best ones of these solutions $h_j$. If one views the number $w$ of weights in $\mathcal{N}$ no longer as a constant, one sees that the number of processores that are needed is simply exponential in $w$, but that the parallel computation time is polynomial in $m$ *and w*.

# Acknowledgements

I would like to thank Peter Auer, Phil Long and Hal White for their helpful comments.

# References

[BR]  A. Blum, R. L. Rivest, "Training a 3-node neural network is NP-complete", *Proc. of the 1988 Workshop on Computational Learning Theory*, Morgan Kaufmann (San Mateo, 1988), 9 - 18

[GJ]  P. Goldberg, M. Jerrum, "Bounding the Vapnik-Chervonenkis dimension of concept classes parameterized by real numbers", *Proc. of the 6th Annual ACM Conference on Computational Learning Theory*, 361 - 369.

[H]  D. Haussler, "Decision theoretic generalizations of the PAC model for neural nets and other learning applications", *Information and Computation*, vol. 100, 1992, 78 - 150

[J]  J. S. Judd, "Neural Network Design and the Complexity of Learning", *MIT-Press* (Cambridge, 1990)

[KV]  M. Kearns, L. Valiant, "Cryptographic limitations on learning boolean formulae and finite automata", *Proc. of the 21st ACM Symposium on Theory of Computing*, 1989, 433 - 444

[KSS]  M. J. Kearns, R. E. Schapire, L. M. Sellie, "Toward efficient agnostic learning", *Proc. of the 5th ACM Workshop on Computational Learning Theory*, 1992, 341 - 352

[M 93a]  W. Maass, "Bounds for the computational power and learning complexity of analog neural nets" (extended abstract), *Proc. of the 25th ACM Symposium on Theory of Computing*, 1993, 335 - 344. Journal version submitted for publication

[M 93b]  W. Maass, "Agnostic PAC-learning of functions on analog neural nets" (journal version), to appear in *Neural Computation*.

[Mi]  J. Milnor, "On the Betti numbers of real varieties", *Proc. of the American Math. Soc.*, vol. 15, 1964, 275 - 280

[PS]  C. H. Papadimitriou, K. Steiglitz, "Combinatorial Optimization: Algorithms and Complexity", Prentice Hall (Englewood Cliffs, 1982)

[V]  L. G. Valiant, "A theory of the learnable", *Comm. of the ACM*, vol. 27, 1984, 1134 - 1142